# Self-Organizing Rules for Robust Principal Component Analysis

**Lei Xu**[1,2]*and **Alan Yuille**[1]
1. Division of Applied Sciences, Harvard University, Cambridge, MA 02138
2. Dept. of Mathematics, Peking University, Beijing, P.R.China

## Abstract

In the presence of outliers, the existing self-organizing rules for *Principal Component Analysis (PCA)* perform poorly. Using statistical physics techniques including the Gibbs distribution, binary decision fields and effective energies, we propose self-organizing PCA rules which are capable of resisting outliers while fulfilling various PCA-related tasks such as obtaining the first principal component vector, the first $k$ principal component vectors, and directly finding the subspace spanned by the first $k$ vector principal component vectors without solving for each vector individually. Comparative experiments have shown that the proposed robust rules improve the performances of the existing PCA algorithms significantly when outliers are present.

## 1   INTRODUCTION

*Principal Component Analysis (PCA)* is an essential technique for data compression and feature extraction, and has been widely used in statistical data analysis, communication theory, pattern recognition and image processing. In the neural network literature, a lot of studies have been made on learning rules for implementing PCA or on networks closely related to PCA (see Xu & Yuille, 1993 for a detailed reference list which contains more than 30 papers related to these issues). The existing rules can fulfil various PCA-type tasks for a number of application purposes.

However, almost all the previously mentioned PCA algorithms are based on the assumption that the data has not been spoiled by outliers (except Xu, Oja&Suen 1992, where outliers can be resisted to some extent.). In practice, real data often contains some outliers and usually they are not easy to separate from the data set. As shown by the experiments described in this paper, these outliers will significantly worsen the performances of the existing PCA learning algorithms. Currently, little attention has been paid to this problem in the neural network literature, although the problem is very important for real applications.

Recently, there have been some success in applying the statistical physics approach to a variety of computer vision problems (Yuille, 1990; Yuille, Yang&Geiger 1990; Yuille, Geiger&Bulthoff, 1991). In particular, it has also been shown that some techniques developed in robust statistics (e.g., redescending M-estimators, least-trimmed squares estimators) appear naturally within the Bayesian formulation by the use of the statistical physics approach. In this paper we adapt this approach to tackle the problem of robust PCA. Robust rules are proposed for various PCA-related tasks such as obtaining the first principal component vector, the first $k$ principal component vectors, and principal subspaces. Comparative experiments have been made and the results show that our robust rules improve the performances of the existing PCA algorithms significantly when outliers are present.

## 2   PCA LEARNING AND ENERGY MINIMIZATION

There exist a number of self-organizing rules for finding the first principal component. Three of them are listed as follows (Oja 1982, 85; Xu, 1991, 93):

$$\vec{m}(t+1) = \vec{m}(t) + \alpha_a(t)(\vec{x}y - \vec{m}(t)y^2), \tag{1}$$

$$\vec{m}(t+1) = \vec{m}(t) + \alpha_a(t)(\vec{x}y - \frac{\vec{m}(t)}{\vec{m}(t)^T\vec{m}(t)}y^2), \tag{2}$$

$$\vec{m}(t+1) = \vec{m}(t) + \alpha_a(t)[y(\vec{x} - \vec{u}) + (y - y')\vec{x}]. \tag{3}$$

where $y = \vec{m}(t)^T\vec{x}$, $\vec{u} = y\vec{m}(t)$, $y' = \vec{m}(t)^T\vec{u}$ and $\alpha_a(t) \geq 0$ is the learning rate which decreases to zero as $t \to \infty$ while satisfying certain conditions, e.g., $\sum_t \alpha_a(t) = \infty$, $\sum_t \alpha_a(t)^q < \infty$ for some $q > 1$.

Each of the three rules will converge to the principal component vector $\vec{\phi}$ almost surely under some mild conditions which are studied in detail in by Oja (1982&85) and Xu (1991&93). Regarding $\vec{m}$ as the weight vector of a linear neuron with output $y = \vec{m}^T\vec{x}$, all the three rules can be considered as modifications of the well known Hebbian rule $\vec{m}(t+1) = \vec{m}(t) + \alpha_a(t)\vec{x}y$ through introducing additional terms for preventing $\|\vec{m}(t)\|$ from going to $\infty$ as $t \to \infty$.

The performances of these rules deteriorate considerably when data contains outliers. Although some outlier-resisting versions of eq.(1) and eq.(2) have also been recently proposed (Xu, Oja & Suen, 1992), they work well only for data which is not severely spoiled by outliers. In this paper, we adopt a totally different approach—we generalize eq.(1),eq.(2) and eq.(3) into more robust versions by using the statistical physics approach.

To do so, first we need to connect these rules to energy functions. It follows from Xu (1991&93) and Xu & Yuille(1993) that the rules eq.(2) and eq.(3) are respectively

*on-line* gradient descent rules for minimizing $J_1(\vec{m})$, $J_2(\vec{m})$ respectively[1]:

$$J_1(\vec{m}) = \frac{1}{N} \sum_{i=1}^{N} (\vec{x}_i^T \vec{x}_i - \frac{\vec{m}^T \vec{x}_i \vec{x}_i^T \vec{m}}{\vec{m}^T \vec{m}}) \tag{4}$$

$$J_2(\vec{m}) = \frac{1}{N} \sum_{i=1}^{N} \|\vec{x}_i - \vec{u}_i\|^2. \tag{5}$$

It has also been proved that the rule given by eq.(1) satisfies (Xu, 1991, 93): (a) $\vec{h}_1^T \vec{h}_2 \geq 0, E(\vec{h}_1)^T L(\vec{h}_1) \geq 0$, with $\vec{h}_1 = \vec{x}y - \vec{m}y^2$, $\vec{h}_2 = \vec{x}y - \frac{\vec{m}}{\vec{m}^T \vec{m}} y^2$; (b) $E(\vec{h}_1)^T E(\vec{h}_3) \geq 0$, with $\vec{h}_3 = y(\vec{x} - \vec{u}) + (y - y')\vec{x}$; (c) Both $J_1$ and $J_2$ have only one local (also global) minimum $tr(\Sigma) - \vec{\phi}^T \Sigma \vec{\phi}$, and all the other critical points (i.e., the points satisfy $\frac{\partial J_i(\vec{m})}{\partial \vec{m}} = 0, i = 1, 2$) are saddle points. Here $\Sigma = E\{\vec{x}\vec{x}^t\}$, and $\vec{\phi}$ is the eigenvector of $\Sigma$ corresponding to the largest eigenvalue.

That is, the rule eq.(1) is a downhill algorithm for minimizing $J_1$ in both the *on line* sense and the *average* sense, and for minimizing $J_2$ in the *average* sense.

## 3   GENERALIZED ENERGY AND ROBUST PCA

We further regard $J_1(\vec{m})$, $J_2(\vec{m})$ as special cases of the following general energy:

$$J(\vec{m}) = \frac{1}{N} \sum_{i=1}^{N} z(\vec{x}_i, \vec{m}), \quad z(\vec{x}_i, \vec{m}) \geq 0. \tag{6}$$

where $z(\vec{x}_i, \vec{m})$ is the portion of energy contributed by the sample $\vec{x}_i$, and

$$\begin{aligned} z(\vec{x}_i, \vec{m}) = & \quad (\vec{x}_i^T \vec{x}_i - \frac{\vec{m}^T \vec{x}_i \vec{x}_i^T \vec{m}}{\vec{m}^T \vec{m}}) \; for \; J_1, \\ = & \qquad \|\vec{x}_i - \vec{u}_i\|^2 \; for \; J_2 \end{aligned} \tag{7}$$

Following (Yuille, 1990 a& b), we now generalize energy eq.(6) into

$$E(\vec{V}, \vec{m}) = \quad \sum_{i=1}^{N} V_i \; z(\vec{x}_i, \vec{m}) + E_{prior}(\vec{V}) \tag{8}$$

where $\vec{V} = \{V_i, i = 1, \cdots, N\}$ is a binary field $\{V_i\}$ with each $V_i$ being a random variable taking value either 0 or 1. $V_i$ acts as a decision indicator for deciding whether $\vec{x}_i$ is an outlier or a sample. When $V_i = 1$, the portion of energy contributed by the sample $\vec{x}_i$ is taken into consideration; otherwise, it is equivalent to discarding $\vec{x}_i$ as an outlier. $E_{prior}(\vec{V})$ is the *a priori* portion of energy contributed by the *a priori* distribution of $\{V_i\}$. A natural choice is

$$E_{prior}(\vec{V}) = \eta \sum_{i=1}^{N} (1 - V_i) \tag{9}$$

This choice of priori has a natural interpretation: for fixed $\vec{m}$ it is energetically favourable to set $V_i = 1$ (i.e., not regarding $\vec{x}_i$ as an outlier) if $z(\vec{x}_i, \vec{m}) < \sqrt{\eta}$ (i.e.,

the portion of energy contributed by $\vec{x}_i$ is smaller than a prespecified threshold) and to set it to 0 otherwise.

Based on $E(\vec{V}, \vec{m})$, we define a Gibbs distribution (Parisi 1988):

$$P[\vec{V}, \vec{m}] = \frac{1}{Z} e^{-\beta E[\vec{V}, \vec{m}]}, \qquad (10)$$

where $Z$ is the partition function which ensures $\sum_{\vec{V}} \sum_{\vec{m}} P[\vec{V}, \vec{m}] = 1$. Then we compute

$$
\begin{aligned}
P_{margin}(\vec{m}) &= \frac{1}{Z} \sum_{\vec{V}} e^{-\beta \sum_i \{V_i z(\vec{x}_i, \vec{m}) + \eta(1 - V_i)\}} \\
&= \frac{1}{Z} \prod_i \sum_{V_i = \{0,1\}} e^{-\beta \{V_i z(\vec{x}_i, \vec{m}) + \eta(1 - V_i)\}} = \frac{1}{Z_m} e^{-\beta E_{eff}(\vec{m})}. \quad (11)
\end{aligned}
$$

$$Z_m = Z e^{N\beta\eta}, \qquad E_{eff}(\vec{m}) = \frac{-1}{\beta} \sum_i \log\{1 + e^{-\beta\{z(\vec{x}_i, \vec{m}) - \eta\}}\}. \qquad (12)$$

$E_{eff}$ is called the effective energy. Each term in the sum for $E_{eff}$ is approximately $z(\vec{x}_i, \vec{m})$ for small values of $z$ but becomes constant as $z(\vec{x}_i, \vec{m}) \to \infty$. In this way outliers, which are more likely to yield large values of $z(\vec{x}_i, \vec{m})$, are treated differently from samples, and thus the estimation $\vec{m}$ obtained by minimizing $E_{eff}(\vec{m})$ will be robust and able to resist outliers.

$E_{eff}(\vec{m})$ is usually not a convex function and may have many local minima. The statistical physics framework suggests using deterministic annealing to minimize $E_{eff}(\vec{m})$. That is, by the following gradient descent rule eq.(13), to minimize $E_{eff}(\vec{m})$ for small $\beta$ and then track the minimum as $\beta$ increases to infinity (the zero temperature limit):

$$\vec{m}(t+1) = \vec{m}(t) - \alpha_b(t) \sum_i \frac{1}{1 + e^{\beta(z(\vec{x}_i, \vec{m}(t)) - \eta)}} \frac{\partial z(\vec{x}_i, \vec{m}(t))}{\partial \vec{m}(t)}. \qquad (13)$$

More specifically, with $z$'s chosen to correspond to the energies $J_1$ and $J_2$ respectively, we have the following *batch-way* learning rules for robust PCA:

$$\vec{m}(t+1) = \vec{m}(t) + \alpha_b(t) \sum_i \frac{1}{1 + e^{\beta(z(\vec{x}_i, \vec{m}(t)) - \eta)}} (\vec{x}_i y_i - \frac{\vec{m}(t)}{\vec{m}(t)^T \vec{m}(t)} y_i^2), \qquad (14)$$

$$\vec{m}(t+1) = \vec{m}(t) + \alpha_b(t) \sum_i \frac{1}{1 + e^{\beta(z(\vec{x}_i, \vec{m}(t)) - \eta)}} [y_i(\vec{x}_i - \vec{u}_i) + (y_i - y_i')\vec{x}_i]. \qquad (15)$$

For data that comes incrementally or in the *on-line* way, we correspondingly have the following *adaptive or stochastic approximation* versions

$$\vec{m}(t+1) = \vec{m}(t) + \alpha_a(t) \frac{1}{1 + e^{\beta(z(\vec{x}_i, \vec{m}(t)) - \eta)}} (\vec{x}_i y_i - \frac{\vec{m}(t)}{\vec{m}(t)^T \vec{m}(t)} y_i^2), \qquad (16)$$

$$\vec{m}(t+1) = \vec{m}(t) + \alpha_a(t) \frac{1}{1 + e^{\beta(z(\vec{x}_i, \vec{m}(t)) - \eta)}} [y_i(\vec{x}_i - \vec{u}_i) + (y_i - y_i')\vec{x}_i]. \qquad (17)$$

It can be observed that the difference between eq.(2) and eq.(16) or eq.(3) and eq.(17) is that the learning rate $\alpha_a(t)$ has been modified by a multiplicative factor

$$\alpha_m(t) = \frac{1}{1 + e^{\beta(z(\vec{x}_i, \vec{m}(t)) - \eta)}}. \tag{18}$$

which adaptively modifies the learning rate to suit the current input $\vec{x}_i$. This modifying factor has a similar function as that used in Xu, Oja&Suen(1992) for robust line fitting. But the modifying factor eq.(18) is more sophisticated and performs better.

Based on the connection between the rule eq.(1) and $J_1$ or $J_2$, given in sec.2, we can also formally use the modifying factor $\alpha_m(t)$ to turn the rule eq.(1) into the following robust version:

$$\vec{m}(t + 1) = \vec{m}(t) + \alpha_a(t)\frac{1}{1 + e^{\beta(z(\vec{x}_i, \vec{m}(t)) - \eta)}}(\vec{x}_i y_i - \vec{m}(t)y_i^2), \tag{19}$$

## 4   ROBUST RULES FOR $k$ PRINCIPAL COMPONENTS

In a similar way to SGA (Oja, 1992) and GHA (Sanger, 1989) we can generalize the robust rules eq.(19), eq.(16) and eq.(17) into the following general form of robust rules for finding the first $k$ principal components:

$$\vec{m}_j(t + 1) = \vec{m}_j(t) + \alpha_a(t)\frac{1}{1 + e^{\beta(z(\vec{x}_i(j), \vec{m}_j(t)) - \eta)}}\Delta\vec{m}_j(\vec{x}_i(j), \vec{m}_j(t)), \tag{20}$$

$$\vec{x}_i(0) = \vec{x}_i, \quad \vec{x}_i(j + 1) = \vec{x}_i(j) - \sum_{r=1}^{j-1} y_i(r)\vec{m}_r(t), \quad y_i(j) = \vec{m}_j^T(t)\vec{x}_i(j), \tag{21}$$

where $\Delta\vec{m}_j(\vec{x}_i(j), \vec{m}_j(t))$, $z(\vec{x}_i(j), \vec{m}_j(t))$ have four possibilities (Xu & Yuille, 1993). As an example, one of them is given here

$$\Delta\vec{m}_j(\vec{x}_i(j), \vec{m}_j(t)) = (\vec{x}_i(j)y_i(j) - \vec{m}_j(t)y_i(j)^2),$$

$$z(\vec{x}_i(j), \vec{m}_j(t)) = \vec{x}_i(j)^T\vec{x}_i(j) - \frac{y_i(j)^2}{\vec{m}_j(t)^T\vec{m}_j(t)}.$$

In this case, eq.(20) can be regarded as the generalization of GHA (Sanger, 1989).

We can also develop an alternative set of rules for a type of nets with asymmetric lateral weights as used in (Rubner&Schulten, 1990). The rules can also get the first $k$ principal components robustly in the presence of outliers (Xu & Yuille, 1993).

## 5   ROBUST RULES FOR PRINCIPAL SUBSPACE

Let $M = [\vec{m}_1, \cdots, \vec{m}_k]$, $\Phi = [\vec{\phi}_1, \cdots, \vec{\phi}_k]$, $\vec{y} = [y_1, \cdots, y_k]^T$ and $\vec{y} = M^T\vec{x}$, it follows from Oja(1989) and Xu(1991) the rules eq.(1), eq.(3) can be generalized into eq.(22) and eq.(23) respectively:

$$\vec{M}(t + 1) = \vec{M}(t) + \alpha_A(t)(\vec{y}\vec{x}^T - \vec{y}\vec{y}^T M(t)) \tag{22}$$

$$\vec{M}(t+1) = \vec{M}(t) + \alpha_A(t)(\vec{y}(\vec{x}-\vec{u})^T - (\vec{y}-\vec{y'})\vec{x}^T), \quad \vec{u} = M\vec{y}, \quad \vec{y'} = M^T\vec{u} \quad (23)$$

In the case without outliers, by both the rules, the weight matrix $M(t)$ will converge to a matrix $M^\infty$ whose column vectors $m_j^\infty, j = 1, \cdots, k$ span the $k$-dimensional principal subspace (Oja, 1989; Xu, 1991&93), although the vectors are, in general, not equal to the $k$ principal component vectors $\vec{\phi}_j, j = 1, \cdots, k$.

Similar to the previously used procedure, we have the following results:

(1). We can show that eq.(23) is an *on-line* or *stochastic approximation* rule which minimizes the energy $J_3$ in the gradient descent way (Xu, 1991& 93):

$$J_3(\vec{m}) = \frac{1}{N}\sum_{i=1}^{N}\|\vec{x}_i - \vec{u}_i\|^2, \quad \vec{u} = M\vec{y}, \quad \vec{y'} = M^T\vec{u}. \quad (24)$$

and that in the *average sense* the subspace rule eq.(22) is also an *on-line* "down-hill" rule for minimizing the energy function $J_3$.

(2). We can also generalize the non-robust rules eq.(22) and eq.(23) into robust versions by using the statistical physics approach again:

$$\vec{M}(t+1) = \vec{M}(t) + \alpha_A(t)\frac{1}{1+e^{\beta(\|\vec{x}_i-\vec{u}_i\|^2-\eta)}}[\vec{y}_i(\vec{x}_i-\vec{u}_i)^T - (\vec{y}_i-\vec{y'}_i)\vec{x}_i^T], \quad (25)$$

$$\vec{M}(t+1) = \vec{M}(t) + \alpha_A(t)\frac{1}{1+e^{\beta(\|\vec{x}_i-\vec{u}_i\|^2-\eta)}}[\vec{y}_i\vec{x}_i^T - \vec{y}_i\vec{y}_i^T M(t)] \quad (26)$$

## 6    EXAMPLES OF EXPERIMENTAL RESULTS

Let $\vec{x}$ from a population of 400 samples with zero mean. These samples are located on an elliptic ring centered at the origin of $R^3$, with its largest elliptic axis being along the direction $(-1, 1, 0)$, the plane of its other two axes intersecting the $x-y$ plane with an acute angle $(30^o)$. Among the 400 samples, 10 points (only 2.5%) are randomly chosen and replaced by outliers. The obtained data set is shown in Fig.1.

Before the outliers were introduced, either the conventional simple-variance-matrix based approach (i.e., solving $S\vec{\phi} = \lambda\vec{\phi}$, $S = \frac{1}{N}\sum_{i=1}^{N}\vec{x}_i\vec{x}_i^T$) or the unrobust rules eqs.(1)(2)(3) can find the correct 1st principal component vector of this data set.

On the data set contaminated by outliers , shown in Fig.1, the result of the simple-variance-matrix based approach has an angular error of $\vec{\phi}_p$ by $71.04^o$—a result definitely unacceptable. The results of using the proposed robust rules eq.(19), eq.(16) and eq.(17) are shown in Fig.2(a) in comparison with those of their unrobust counterparts— the rules eq.(1), eq.(2) and eq.(3). We observe that all the unrobust rules get the solutions with errors of more than $21^o$ from the correct direction of $\vec{\phi}_p$. By contrast, the robust rules can still maintain a very good accuracy—the error is about $0.36^o$. Fig.2(b) gives the results of solving for the first two principal component vectors. Again, the unrobust rule produce large errors of around $23^o$, while the robust rules have an error of about $1.7^o$. Fig.3 shows the results of solving for the 2-dimensional principal subspace, it is easy to see the significant improvements obtained by using the robust rules.

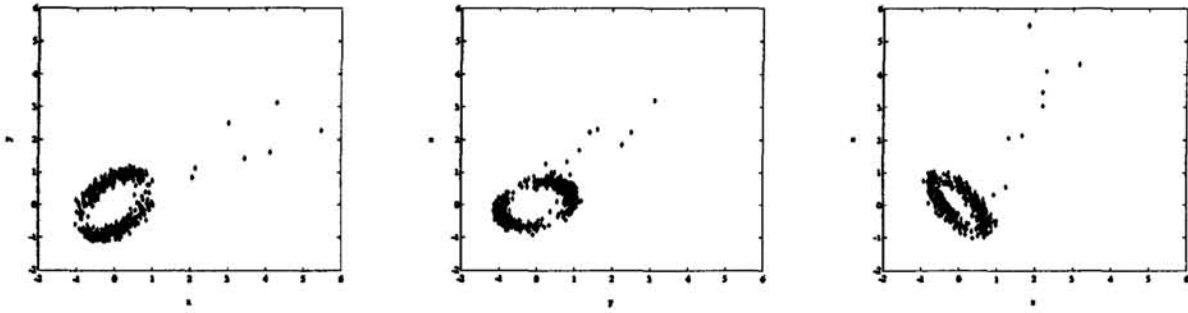

Figure 1: The projections of the data on the $x-y$, $y-z$ and $z-x$ planes, with 10 outliers.

## Acknowledgements

We would like to thank DARPA and the Air Force for support with contracts AFOSR-89-0506 and F4969092-J-0466.

We like to menta ion that some further issues about the proposed robust rules are studied in Xu & Yuille (1993), including the selection of parameters $\alpha$, $\beta$ and $\eta$, the extension of the rules for robust *Minor Component Analysis (MCA)*, the relations between the rules to the two main types of existing robust PCA algorithms in the literature of statistics, as well as to Maximal Likelihood (ML) estimation of finite mixture distributions.

## Footnotes

*Present address: Dept. of Brain and Cognitive Sciences, E10–243, Massachusetts Institute of Technology, Cambridge, MA 02139.

[1]We have $J_1(\vec{m}) \geq 0$, since $\vec{x}^T \vec{x} - \frac{y^2}{\vec{m}^T \vec{m}} = \|\vec{x}\|^2 \sin^2 \theta_{\vec{x}\vec{m}} \geq 0$.

## References

E. Oja, *J. Math. Bio. 16*, 1982, 267-273.

E. Oja & J. Karhunen, *J. Math. Anal. Appl. 106*, 1985, 69-84.

E. Oja, *Int. J. Neural Systems 1*, 1989, 61-68.

E. Oja, *Neural Networks 5*, 1992, 927-935.

G. Parisi, *Statistical Field Theory*, Addison-Wesley, Reading, Mass., 1988.

J. Rubner & K. Schulten, *Biological Cybernetics, 62*, 1990, 193-199.

T.D. Sanger, *Neural Networks, 2*, 1989, 459-473.

L. Xu, *Proc. of IJCNN'91-Singapore*, Nov., 1991, 2368-2373.

L. Xu, Least mean square error reconstruction for self-organizing neural-nets, *Neural Networks 6*, 1993, in press.

L. Xu, E. Oja & C.Y. Suen, *Neural Networks 5*, 1992, 441-457.

L. Xu & A.L. Yuille, Robust principal component analysis by self-organizing rules based on statistical physics approach, *IEEE Trans. Neural Networks*, 1993, in press.

A.L. Yuille, *Neural computation 2*, 1990, 1-24.

A.L. Yuille, D. Geiger and H.H. Bulthoff,*Networks 2*, 1991. 423-442.

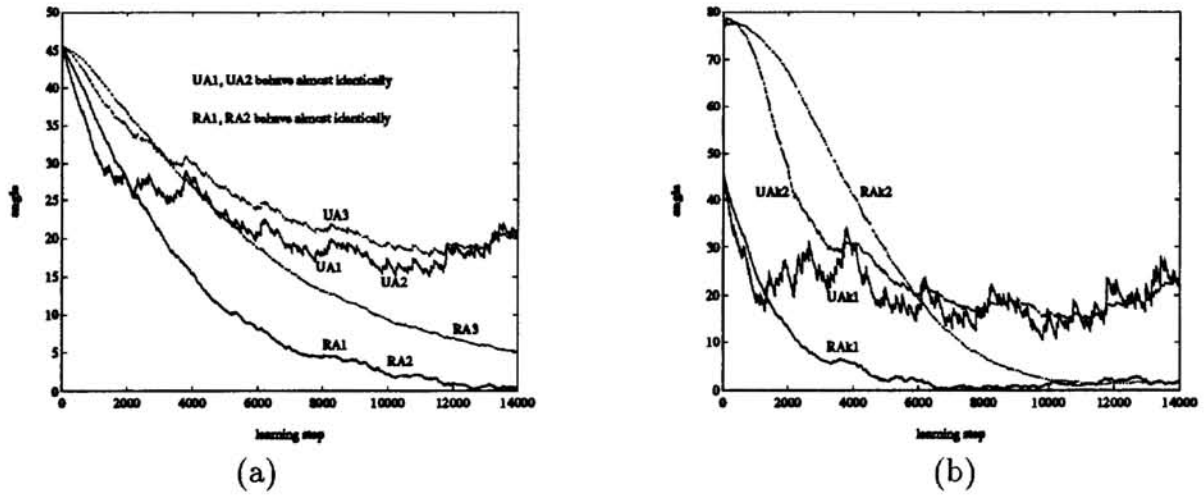

(a)                                            (b)

Figure 2: The learning curves obtained in the comparative experiments for principal component vectors. (a) for the first principal component vector, $RA1$, $RA2$, $RA3$ denote the robust rules eq.(19), eq.(16) and eq.(17) respectively, and $UA1, UA2, UA3$ denote the rules eq.(1), eq.(2) and eq.(3) respectively. The horizontal axis denotes the learning steps, and the vertical axis is $\theta_{\vec{m}(t)\vec{\phi}_{p1}}$ with $\theta_{\vec{x},\vec{y}}$ denoting the acute angle between $\vec{x}$ and $\vec{y}$. (b) for the first two principal component vectors, by the robust rule eq.(20) and its unrobust counterpart GHA. $UAk1$, $UAk2$ denote the learning curves of angles $\theta_{\vec{m}_1(t)\vec{\phi}_{p1}}$ and $\theta_{\vec{m}_2(t)\vec{\phi}_{p2}}$ respectively, obtained by GHA. $RAk1$, $RAk2$ denote the learning curves of the angles obtained by using the robust rule eq.(20). In both (a) & (b), $\vec{\phi}_{pj}, j = 1, 2$ is the correct 1st and 2nd principal component vector respectively.

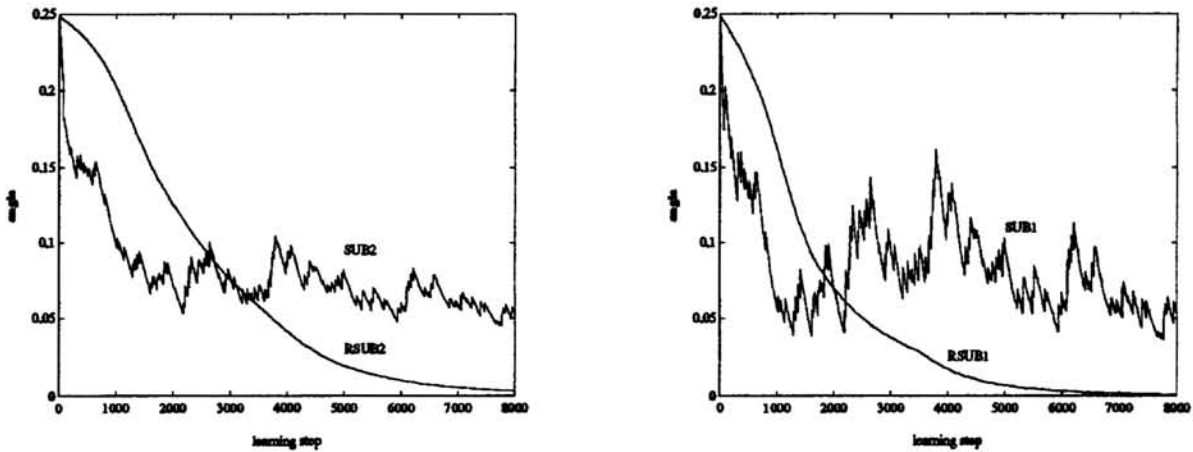

Figure 3: The learning curves obtained in the comparative experiments for for solving the 2-dimensional principal subspace. Each learning curve expresses the change of the residual $e_r(t) = \sum_{j=1}^{2} \|\vec{m}_j(t) - \sum_{r=1}^{2} (\vec{m}_j(t)^T \vec{\phi}_{pr})\vec{\phi}_{pr}\|^2$ with learning steps. The smaller the residual, the closer the estimated principal subspace to the correct one. $SUB1$, $SUB2$ denote the unrobust rules eq.(22) and eq.(23) respectively, and $RSUB1$, $RSUB2$ denote the robust rules eq.(26) and eq.(25) respectively.